# Relax then Compensate:
# On Max-Product Belief Propagation and More

**Arthur Choi**
Computer Science Department
University of California, Los Angeles
Los Angeles, CA 90095
aychoi@cs.ucla.edu

**Adnan Darwiche**
Computer Science Department
University of California, Los Angeles
Los Angeles, CA 90095
darwiche@cs.ucla.edu

## Abstract

We introduce a new perspective on approximations to the maximum a posteriori (MAP) task in probabilistic graphical models, that is based on simplifying a given instance, and then tightening the approximation. First, we start with a structural relaxation of the original model. We then infer from the relaxation its deficiencies, and compensate for them. This perspective allows us to identify two distinct classes of approximations. First, we find that max-product belief propagation can be viewed as a way to compensate for a relaxation, based on a particular idealized case for exactness. We identify a second approach to compensation that is based on a more refined idealized case, resulting in a new approximation with distinct properties. We go on to propose a new class of algorithms that, starting with a relaxation, iteratively seeks tighter approximations.

## 1 Introduction

Relaxations are a popular approach for tackling intractable optimization problems. Indeed, for finding the maximum a posteriori (MAP) assignment in probabilistic graphical models, relaxations play a key role in a variety of algorithms. For example, tree-reweighted belief propagation (TRW-BP) can be thought of as a linear programming relaxation of an integer program for a given MAP problem [1, 2]. Branch-and-bound search algorithms for finding optimal MAP solutions, such as [3, 4], rely on structural relaxations, such as mini-bucket approximations, to provide upper bounds [4, 5].

Whether a relaxation is used as an approximation on its own, or as a guide for finding optimal solutions, a trade-off is typically made between the quality of an approximation and the complexity of computing it. We illustrate here instead how it is possible to tighten a given relaxation itself, without impacting its structural complexity.

More specifically, we propose here an approach to approximating a given MAP problem by performing two steps. First, we *relax* the structure of a given probabilistic graphical model, which results in a simpler model whose MAP solution provides an upper bound on that of the original. Second, we *compensate* for the relaxation by introducing auxiliary parameters, which we use to restore certain properties, leading to a tighter approximation. We shall in fact propose two distinct properties on which a compensation can be based. The first is based on a simplified case where a compensation can be guaranteed to yield exact results. The second is based on a notion of an ideal compensation, that seeks to correct for a relaxation more directly. As we shall see, the first approach leads to a new semantics for the max-product belief propagation algorithm. The second approach leads to another approximation that further yields upper bounds on the MAP solution. We further propose an algorithm for finding such a compensation, that starts with a relaxation and iteratively provides monotonically decreasing upper bounds on the MAP solution (at least empirically).

Proofs of results are given in the auxiliary Appendix.

## 2   MAP Assignments

Let $\mathcal{M}$ be a factor graph over a set of variables $\mathbf{X}$, inducing a distribution $Pr(\mathbf{x}) \propto \prod_a \psi_a(\mathbf{x}_a)$ where $\mathbf{x} = \{X_1 = x_1, \dots, X_n = x_n\}$ is an assignment of factor graph variables $X_i$ to states $x_i$, and where $a$ is an index to the factor $\psi_a(\mathbf{X}_a)$ over the domain $\mathbf{X}_a \subseteq \mathbf{X}$. We seek the *maximum a posteriori* (MAP) assignment $\mathbf{x}^\star = \operatorname{argmax}_{\mathbf{x}} \prod_a \psi_a(\mathbf{x}_a)$. We denote the log of the value of a MAP assignment $\mathbf{x}^\star$ by:

$$\mathsf{map}^\star = \log \max_{\mathbf{x}} \prod_a \psi_a(\mathbf{x}_a) = \max_{\mathbf{x}} \sum_a \log \psi_a(\mathbf{x}_a)$$

which we refer to more simply as the MAP value. Note that there may be multiple MAP assignments $\mathbf{x}^\star$, so we may refer to just the value $\mathsf{map}^\star$ when the particular assignment is not relevant. Next, if $\mathbf{z}$ is an assignment over variables $\mathbf{Z} \subseteq \mathbf{X}$, then let $\mathbf{x} \sim \mathbf{z}$ denote that $\mathbf{x}$ and $\mathbf{z}$ are compatible assignments, i.e., they set their common variables to the same states. Consider then the MAP value under a partial assignment $\mathbf{z}$:

$$\mathsf{map}(\mathbf{z}) = \max_{\mathbf{x} \sim \mathbf{z}} \sum_a \log \psi_a(\mathbf{x}_a).$$

We will, in particular, be interested in the MAP value $\mathsf{map}(X = x)$ where we assume a single variable $X$ is set to a particular state $x$. We shall also refer to these MAP values more generally as $\mathsf{map}(.)$, without reference to any particular assignment.

## 3   Relaxation

The structural relaxations that we consider here are based on the relaxation of *equivalence constraints* from a model $\mathcal{M}$, where an equivalence constraint $X_i \equiv X_j$ is a factor $\psi_{eq}(X_i, X_j)$ over two variables $X_i$ and $X_j$ that have the same states. Further, $\psi_{eq}(x_i, x_j)$ is 1 if $x_i = x_j$ and 0 otherwise. We call an assignment $\mathbf{x}$ *valid*, with respect to an equivalence constraint $X_i \equiv X_j$, if it sets variables $X_i$ and $X_j$ to the same state, and *invalid* otherwise. Note that when we remove an equivalence constraint from a model $\mathcal{M}$, the values $\mathsf{map}(\mathbf{x})$ for valid configurations $\mathbf{x}$ do not change, since $\log 1 = 0$. However, the values $\mathsf{map}(\mathbf{x})$ for invalid configurations can increase, since they are $-\infty$ prior to the removal. In fact, they could overtake the optimal value $\mathsf{map}^\star$. Thus, the MAP value after relaxing an equivalence constraint in $\mathcal{M}$ is an upper bound on the original MAP value.

It is straightforward to augment a model $\mathcal{M}$ to another where equivalence constraints can be relaxed. Consider, for example, a factor $\psi_1(A, B, C)$. We can replace the variable $C$ in this factor with a clone variable $C'$, resulting in a factor $\psi_1'(A, B, C')$. When we now add the factor $\psi_2(C, C')$ for the equivalence constraint $C \equiv C'$, we have a new model $\mathcal{M}'$ which is equivalent to the original model $\mathcal{M}$, in that an assignment $\mathbf{x}$ in $\mathcal{M}$ corresponds to an assignment $\mathbf{x}'$ in $\mathcal{M}'$, where assignment $\mathbf{x}'$ sets a variable and its clone to the same state. Moreover, the value $\mathsf{map}(\mathbf{x})$ in model $\mathcal{M}$ is the same as the value $\mathsf{map}'(\mathbf{x}')$ in model $\mathcal{M}'$.

We note that a number of structural relaxations can be reduced to the removal of equivalence constraints, including relaxations found by deleting edges [6, 7], as well as mini-bucket approximations [5, 4]. In fact, the example above can be considered a relaxation where we delete a factor graph edge $C \to \psi_1$, substituting clone $C'$ in place of variable $C$. Note that mini-bucket approximations in particular have enabled algorithms for solving MAP problems via branch-and-bound search [3, 4].

## 4   Compensation

Suppose that we have a model $\mathcal{M}$ with MAP values $\mathsf{map}(.)$. Say that we remove the equivalence constraints in $\mathcal{M}$, resulting in a relaxed model with MAP values $\mathsf{r\text{-}map}(.)$. Our goal is to identify a compensated model $\mathcal{M}'$ with MAP values $\mathsf{c\text{-}map}(.)$ that is as tractable to compute as the values $\mathsf{r\text{-}map}(.)$, but yielding tighter approximations of the original values $\mathsf{map}(.)$.

To this end, we introduce into the relaxation additional factors $\psi_{ij;i}(X_i)$ and $\psi_{ij;j}(X_j)$ for each equivalence constraint $X_i \equiv X_j$ that we remove. Equivalently, we can introduce the log factors $\theta(X_i) = \log \psi_{ij;i}(X_i)$ and $\theta(X_j) = \log \psi_{ij;j}(X_j)$ (we omit the additional factor indices, as they

will be unambiguous from the context). These new factors add new parameters into the approximation, which we shall use to recover a weaker notion of equivalence into the model. More specifically, given a set of equivalence constraints $X_i \equiv X_j$ to relax, we have the original MAP values $\mathsf{map}(.)$, the relaxation $\mathsf{r\text{-}map}(.)$ and the compensation $\mathsf{c\text{-}map}(.)$, where:

- $\mathsf{map}(\mathbf{z}) = \max_{\mathbf{x} \sim \mathbf{z}} \sum_a \log \psi_a(\mathbf{x}_a) + \sum_{X_i \equiv X_j} \log \psi_{eq}(X_i = x_i, X_j = x_j)$
- $\mathsf{r\text{-}map}(\mathbf{z}) = \max_{\mathbf{x} \sim \mathbf{z}} \sum_a \log \psi_a(\mathbf{x}_a)$
- $\mathsf{c\text{-}map}(\mathbf{z}) = \max_{\mathbf{x} \sim \mathbf{z}} \sum_a \log \psi_a(\mathbf{x}_a) + \sum_{X_i \equiv X_j} \theta(X_i = x_i) + \theta(X_j = x_j)$

Note that the auxiliary factors $\theta$ of the compensation do not introduce additional complexity to the relaxation, in the sense that the treewidth of the resulting model is the same as that of the relaxation.

Consider then the case where an optimal assignment $\mathbf{x}^\star$ for the relaxation happens to set variables $X_i$ and $X_j$ to the same state $x$, for each equivalence constraint $X_i \equiv X_j$ that we relaxed. In this case, the optimal solution for the relaxation is also an optimal solution for the original model, i.e., $\mathsf{r\text{-}map}^\star = \mathsf{map}^\star$. On the other hand, if a relaxation's optimal assignment sets $X_i$ and $X_j$ to different states, then it is not a valid assignment for the original model $\mathcal{M}$, as it violates the equivalence constraint and thus has log probability $-\infty$.

Consider, for a given equivalence constraint $X_i \equiv X_j$, the relaxation's MAP values $\mathsf{r\text{-}map}(X_i = x)$ and $\mathsf{r\text{-}map}(X_j = x)$ when we set, respectively, a single variable $X_i$ or $X_j$ to a state $x$. If for all states $x$ we find that $\mathsf{r\text{-}map}(X_i = x) \neq \mathsf{r\text{-}map}(X_j = x)$, then we can infer that the MAP assignment sets variables $X_i$ and $X_j$ to different states: the MAP value when we set $X_i$ to a state $x$ is different than the MAP value when we set $X_j$ to the same state. We can then ask of a compensation, for all states $x$, that $\mathsf{c\text{-}map}(X_i = x) = \mathsf{c\text{-}map}(X_j = x)$, enforcing a weaker notion of equivalence. In this case, if there is a MAP assignment that sets variable $X_i$ to a state $x$, then there is at least a MAP assignment that sets variable $X_j$ to the same state, even if there is no MAP assignment that sets both $X_i$ and $X_j$ to the same state at the same time.

We now want to identify parameters $\theta(X_i)$ and $\theta(X_j)$ to compensate for a relaxation in this manner. We propose two approaches: (1) based on a condition for exactness in a special case, and (2) based on a notion of ideal compensations. To get the intuitions behind these approaches, we consider first the simplified case where a single equivalence constraint is relaxed.

## 4.1 Intuitions: Splitting a Model into Two

Consider the case where relaxing a single equivalence constraint $X_i \equiv X_j$ splits a model $\mathcal{M}$ into two independent sub-models, $\mathcal{M}_i$ and $\mathcal{M}_j$, where sub-model $\mathcal{M}_i$ contains variable $X_i$ and sub-model $\mathcal{M}_j$ contains variable $X_j$. Intuitively, we would like the parameters added in one sub-model to summarize the relevant information about the other sub-model. In this way, each sub-model could independently identify their optimal sub-assignments. For example, we can use the parameters:

$$\theta(X_i = x) = \mathsf{map}_j(X_j = x) \quad \text{and} \quad \theta(X_j = x) = \mathsf{map}_i(X_i = x).$$

Since sub-models $\mathcal{M}_i$ and $\mathcal{M}_j$ become independent after relaxing the single equivalence constraint $X_i \equiv X_j$, computing these parameters is sufficient to reconstruct the MAP solution for the original model $\mathcal{M}$. In particular, we have that $\theta(X_i = x) + \theta(X_j = x) = \mathsf{map}(X_i = x, X_j = x)$, and further that $\mathsf{map}^\star = \max_x [\theta(X_i = x) + \theta(X_j = x)]$.

We propose then that the parameters of a compensation, with MAP values $\mathsf{c\text{-}map}(.)$, should satisfy the following condition:

$$\mathsf{c\text{-}map}(X_i = x) \quad = \quad \mathsf{c\text{-}map}(X_j = x) \quad = \quad \theta(X_i = x) + \theta(X_j = x) + \gamma \qquad (1)$$

for all states $x$. Here $\gamma$ is an arbitrary normalization constant, but the choice $\gamma = \frac{1}{2}\mathsf{c\text{-}map}^\star$ results in simpler semantics. The following proposition confirms that this choice of parameters does indeed reflect our earlier intuitions, showing that this choice allows us to recover exact solutions in the idealized case when a model is split into two.

**Proposition 1** *Let* $\mathsf{map}(.)$ *denote the MAP values of a model* $\mathcal{M}$, *and let* $\mathsf{c\text{-}map}(.)$ *denote the MAP values of a compensation that results from relaxing an equivalence constraint* $X_i \equiv X_j$ *that split* $\mathcal{M}$ *into two independent sub-models. Then the compensation has parameters satisfying Equation 1 iff* $\mathsf{c\text{-}map}(X_i = x) = \mathsf{c\text{-}map}(X_j = x) = \mathsf{map}(X_i = x, X_j = x) + \gamma$.

Note that the choice $\gamma = \frac{1}{2}$c-map$^\star$ implies that $\theta(X_i\!=\!x) + \theta(X_j\!=\!x) = \mathsf{map}(X_i\!=\!x, X_j\!=\!x)$ in the case where relaxing an equivalent constraint splits a model into two.

In the case where relaxing an equivalence constraint does not split a model into two, a compensation satisfying Equation 1 at least satisfies a weaker notion of equivalence. We might expect that such a compensation may lead to more meaningful, and hopefully more accurate, approximations than a relaxation. Indeed, this compensation will eventually lead to a generalized class of belief propagation approximations. Thus, we call a compensation satisfying Equation 1 a REC-BP approximation.

## 4.2 Intuitions: An Ideal Compensation

In the case where a single equivalence constraint $X_i \equiv X_j$ is relaxed, we may imagine the possibility of an "ideal" compensation where, as far as computing the MAP solution is concerned, a compensated model is as good as a model where the equivalence constraint was not relaxed. Consider then the following proposal of an ideal compensation, which has the following two properties. First, it has *valid configurations*:

$$\mathsf{c\text{-}map}(X_i\!=\!x) \quad = \quad \mathsf{c\text{-}map}(X_j\!=\!x) \quad = \quad \mathsf{c\text{-}map}(X_i\!=\!x, X_j\!=\!x)$$

for all states $x$. Second it has *scaled values* for valid configurations:

$$\mathsf{c\text{-}map}(X_i\!=\!x, X_j\!=\!x) \quad = \quad \kappa \cdot \mathsf{map}(X_i\!=\!x, X_j\!=\!x).$$

for all states $x$, and for some $\kappa > 1$. If a compensation has valid configurations, then its optimal solution sets variables $X_i$ and $X_j$ to the same state, and is thus a valid assignment for the original instance (it satisfies the equivalence constraint). Moreover, if it has scaled values, then the compensation further allows us to recover the MAP value as well. A compensation having valid configurations and scaled values is thus ideal as it is sufficient for us to recover the exact solution.

It may not always be possible to find parameters that lead to an ideal compensation. However, we propose that a compensation's parameters should satisfy:

$$\mathsf{c\text{-}map}(X_i\!=\!x) \quad = \quad \mathsf{c\text{-}map}(X_j\!=\!x) \quad = \quad 2 \cdot [\theta(X_i\!=\!x) + \theta(X_j\!=\!x)] \qquad (2)$$

for all states $x$, where we choose $\kappa = 2$. As the following proposition tells us, if a compensation is an ideal one, then it must at least satisfy Equation 2.

**Proposition 2** *Let* $\mathsf{map}(.)$ *denote the MAP values of a model* $\mathcal{M}$*, and let* $\mathsf{c\text{-}map}(.)$ *denote the MAP values of a compensation that results from relaxing an equivalence constraint* $X_i \equiv X_j$ *in* $\mathcal{M}$*. If* $\mathsf{c\text{-}map}(.)$ *has valid configurations and scaled values, then* $\mathsf{c\text{-}map}(.)$ *satisfies Equation 2.*

We thus call a compensation satisfying Equation 2 a REC-I compensation.

We note that other values of $\kappa > 1$ can be used, but the choice $\kappa = 2$ given above results in simpler semantics. In particular, if a compensation happens to satisfy $\mathsf{c\text{-}map}(X_i\!=\!x) = \mathsf{c\text{-}map}(X_j\!=\!x) = \mathsf{c\text{-}map}(X_i\!=\!x, X_j\!=\!x)$ for some state $x$, we have that $\theta(X_i\!=\!x) + \theta(X_j\!=\!x) = \mathsf{map}(X_i\!=\!x, X_j\!=\!x)$ (i.e., the parameters alone can recover an original MAP value).

Before we discuss the general case where we relax multiple equivalence constraints, we highlight first a few properties shared by both REC-BP and REC-I compensations, that shall follow from more general results that we shall present. First, if the optimal assignment $\mathbf{x}^\star$ for a compensation sets the variables $X_i$ and $X_j$ to the same state, then: (1) the assignment $\mathbf{x}^\star$ is also optimal for the original model $\mathcal{M}$; and (2) $\frac{1}{2}$c-map$^\star$ = map$^\star$. In the case where $\mathbf{x}^\star$ does not set variables $X_i$ and $X_j$ to the same state, the value c-map$^\star$ gives at least an upper bound that is no worse than the bound given by the relaxation alone. In particular:

$$\mathsf{map}^\star \leq \frac{1}{2}\mathsf{c\text{-}map}^\star \leq \mathsf{r\text{-}map}^\star.$$

Thus, at least in the case where a single equivalence constraint is relaxed, the compensations implied by Equations 1 and 2 do indeed tighten a relaxation (see the auxiliary Appendix for further details).

## 4.3 General Properties

In this section, we identify the conditions that compensations should satisfy in the more general case where multiple equivalence constraints are relaxed, and further highlight some of their properties.

Suppose that $k$ equivalence constraints $X_i \equiv X_j$ are relaxed from a given model $\mathcal{M}$. Then compensations REC-BP and REC-I seek to recover into the relaxation two weaker notions of equivalence.

First, a REC-BP compensation has auxiliary parameters satisfying:

$$\mathsf{c\text{-}map}(X_i\!=\!x) \quad = \quad \mathsf{c\text{-}map}(X_j\!=\!x) \quad = \quad \theta(X_i\!=\!x) + \theta(X_j\!=\!x) + \gamma \qquad (3)$$

where $\gamma = \frac{k}{1+k}\mathsf{c\text{-}map}^\star$. We then approximate the exact MAP value $\mathsf{map}^\star$ by the value $\frac{1}{1+k}\mathsf{c\text{-}map}^\star$.

The following theorem relates REC-BP to max-product belief propagation.

**Theorem 1** *Let $\mathsf{map}(.)$ denote the MAP values of a model $\mathcal{M}$, and let $\mathsf{c\text{-}map}(.)$ denote the MAP values of a compensation that results from relaxing enough equivalence constraints $X_i \equiv X_j$ in $\mathcal{M}$ to render it fully disconnected. Then a compensation whose parameters satisfy Equation 3 has values $\exp\{\mathsf{c\text{-}map}(X_i\!=\!x)\}$ that correspond to the max-marginals of a fixed-point of max-product belief propagation run on $\mathcal{M}$, and vice-versa.*

Loopy max-product belief propagation is thus the degenerate case of a REC-BP compensation, when the approximation is fully disconnected (by deleting every factor graph edge, as defined in Section 3). Approximations need not be this extreme, and more structured approximations correspond to instances in the more general class of iterative joingraph propagation approximations [8, 6].

Next, a REC-I compensation has parameters satisfying:

$$\mathsf{c\text{-}map}(X_i\!=\!x) \quad = \quad \mathsf{c\text{-}map}(X_j\!=\!x) \quad = \quad (1+k)[\theta(X_i\!=\!x) + \theta(X_j\!=\!x)] \qquad (4)$$

We again approximate the exact MAP value $\mathsf{map}^\star$ with the value $\frac{1}{1+k}\mathsf{c\text{-}map}^\star$.

In both compensations, it is possible to determine if the optimal assignment $\mathbf{x}^\star$ of a compensation is an optimal assignment for the original model $\mathcal{M}$: we need only check that it is a valid assignment.

**Theorem 2** *Let $\mathsf{map}(.)$ denote the MAP values of a model $\mathcal{M}$, and let $\mathsf{c\text{-}map}(.)$ denote the MAP values of a compensation that results from relaxing $k$ equivalence constraints $X_i \equiv X_j$. If the compensation has parameters satisfying either Eqs. 3 or 4, and if $\mathbf{x}^\star$ is an optimal assignment for the compensation that is also valid, then: (1) $\mathbf{x}^\star$ is optimal for the model $\mathcal{M}$, and (2) $\frac{1}{1+k}\mathsf{c\text{-}map}^\star = \mathsf{map}^\star$.*

This result is analogous to results for max-product BP, TRW-BP, and related algorithms [9, 2, 10].

A REC-I compensation has additional properties over a REC-BP compensation. First, a REC-I compensation yields upper bounds on the MAP value, whereas REC-BP does not yield a bound in general.

**Theorem 3** *Let $\mathsf{map}(.)$ denote the MAP values of a model $\mathcal{M}$, and let $\mathsf{c\text{-}map}(.)$ denote the MAP values of a compensation that results from relaxing $k$ equivalence constraints $X_i \equiv X_j$. If the compensation has parameters satisfying Equation 4, then $\mathsf{map}^\star \leq \frac{1}{1+k}\mathsf{c\text{-}map}^\star$.*

We remark now that a relaxation alone has analogous properties. If an assignment $\mathbf{x}^\star$ is optimal for a relaxation with MAP values $\mathsf{r\text{-}map}(.)$, and it is also a valid assignment for a model $\mathcal{M}$ (i.e., it does not violate the equivalence constraints $X_i \equiv X_j$), then $\mathbf{x}^\star$ is also optimal for $\mathcal{M}$, where $\mathsf{r\text{-}map}(\mathbf{x}^\star) = \mathsf{map}(\mathbf{x}^\star)$ (since they are composed of the same factor values). If an assignment $\mathbf{x}^\star$ of a relaxation is not valid for model $\mathcal{M}$, then the MAP value of the relaxation is an upper bound on the original MAP value. On the other hand, REC-I compensations are tighter approximations than the corresponding relaxation, at least in the case when a single equivalence constraint is relaxed: $\mathsf{map}^\star \leq \frac{1}{2}\mathsf{c\text{-}map}^\star \leq \mathsf{r\text{-}map}^\star$. When we relax multiple equivalence constraints we find, at least empirically, that REC-I bounds are never worse than relaxations, although we leave this point open.

The following theorem has implications for MAP solvers that rely on relaxations for upper bounds.

**Theorem 4** *Let $\mathsf{map}(.)$ denote the MAP values of a model $\mathcal{M}$, and let $\mathsf{c\text{-}map}(.)$ denote the MAP values of a compensation that results from relaxing $k$ equivalence constraints $X_i \equiv X_j$. If the compensation has parameters satisfying Eq. 4, and if $\mathbf{z}$ is a partial assignment that sets the same sign to variables $X_i$ and $X_j$, for any equivalence constraint $X_i \equiv X_j$ relaxed, then: $\mathsf{map}(\mathbf{z}) \leq \frac{1}{1+k}\mathsf{c\text{-}map}(\mathbf{z})$.*

Algorithms, such as those in [3, 4], perform a depth-first branch-and-bound search to find an optimal MAP solution. They rely on upper bounds of a MAP solution, under partial assignments, in order to prune the search space. Thus, any method capable of providing upper bounds tighter than those of a relaxation, can potentially have an impact in the performance of a branch-and-bound MAP solver.

**Algorithm 1** RelaxEq-and-Compensate (REC)

**input:** a model $\mathcal{M}$ with $k$ equivalence constraints $X_i \equiv X_j$
**output:** a compensation $\mathcal{M}'$
**main:**
1: $\mathcal{M}'_0 \leftarrow$ result of relaxing all $X_i \equiv X_j$ in $\mathcal{M}$
2: add to $\mathcal{M}'_0$ the factors $\theta(X_i), \theta(X_j)$, for each $X_i \equiv X_j$
3: initialize all parameters $\theta_0(X_i\!=\!x), \theta_0(X_j\!=\!x)$, e.g., to $\frac{1}{2}$r-map$^\star$
4: $t \leftarrow 0$
5: **while** parameters have not converged **do**
6:     $t \leftarrow t + 1$
7:     **for** each equivalence constraint $X_i \equiv X_j$ **do**
8:        update parameters $\theta(X_i\!=\!x)^t, \theta(X_j\!=\!x)^t$, computed using compensation $\mathcal{M}'_{t-1}$, by:
9:          for REC-BP: Equations 5 & 6
10:        for REC-I: Equations 7 & 8
11:     $\theta^t(X_i) \leftarrow q \cdot \theta^t(X_i) + (1-q) \cdot \theta^{t-1}(X_i)$ and $\theta^t(X_j) \leftarrow q \cdot \theta^t(X_j) + (1-q) \cdot \theta^{t-1}(X_j)$
12: **return** $\mathcal{M}'_t$

## 5 An Algorithm to Find Compensations

Up to this point, we have not discussed how to actually find the auxiliary parameters $\theta(X_i\!=\!x)$ and $\theta(X_j\!=\!x)$ of a compensation. However, Equations 3 and 4 naturally suggest iterative algorithms for finding REC-BP and REC-I compensations. Consider, for the case of REC-BP, the fact that parameters satisfy Equation 3 iff they satisfy:

$$
\begin{aligned}
\theta(X_i\!=\!x) &= \text{c-map}(X_j\!=\!x) - \theta(X_j\!=\!x) - \gamma \\
\theta(X_j\!=\!x) &= \text{c-map}(X_i\!=\!x) - \theta(X_i\!=\!x) - \gamma
\end{aligned}
$$

This suggests an iterative fixed-point procedure for finding the parameters of a compensation that satisfy Equation 3. First, we start with an initial compensation with MAP values c-map$_0(.)$, where parameters have been initialized to some value. For an iteration $t > 0$, we can update our parameters using the compensation from the previous iteration:

$$
\begin{aligned}
\theta_t(X_i\!=\!x) &= \text{c-map}_{t-1}(X_j\!=\!x) - \theta_{t-1}(X_j\!=\!x) - \gamma_{t-1} & (5) \\
\theta_t(X_j\!=\!x) &= \text{c-map}_{t-1}(X_i\!=\!x) - \theta_{t-1}(X_i\!=\!x) - \gamma_{t-1} & (6)
\end{aligned}
$$

where $\gamma_{t-1} = \frac{k}{1+k}\text{c-map}^\star_{t-1}$. If at some point, the parameters of one iteration do not change in the next, then we can say that the iterations have converged, and that the compensation satisfies Equation 3. Similarly, for REC-I compensations, we use the update equations:

$$
\begin{aligned}
\theta_t(X_i\!=\!x) &= \tfrac{1}{1+k}\text{c-map}_{t-1}(X_j\!=\!x) - \theta_{t-1}(X_j\!=\!x) & (7) \\
\theta_t(X_j\!=\!x) &= \tfrac{1}{1+k}\text{c-map}_{t-1}(X_i\!=\!x) - \theta_{t-1}(X_i\!=\!x) & (8)
\end{aligned}
$$

to identify compensations that satisfy Equation 4.

Algorithm 1 summarizes our proposal to compensate for a relaxation, using the iterative procedures for REC-BP and REC-I. We refer to this algorithm more generically as RelaxEq-and-Compensate (REC). Note that in Line 11, we further damp the updates by $q$, which is typical for such algorithms (we use $q = \frac{1}{2}$). Note also that in Line 3, we suggest that we initialize parameters by $\frac{1}{2}$r-map$^\star$. The consequence of this is that our initial compensation has the MAP value $\frac{1}{1+k}\text{c-map}^\star_0 = $ r-map$^\star$.[1] That is, the initial compensation is equivalent to the relaxation, for both REC-BP and REC-I. Typically, both algorithms tend to have compensations with decreasing MAP values. REC-BP may eventually have MAP values that oscillate however, and may not converge. On the other hand, by Theorem 3, we know that a REC-I compensation must yield an upper bound on the true MAP value map$^\star$. Starting with an initial upper bound r-map$^\star$ from the relaxation, REC-I yields, at least empirically, monotonically decreasing upper bounds on the true MAP value from iteration to iteration. We explore this point further in the following section.

$= \max_\mathbf{x}[\text{r-map}(\mathbf{x}) + k \cdot \text{r-map}^\star] = \text{r-map}^\star + k \cdot \text{r-map}^\star$

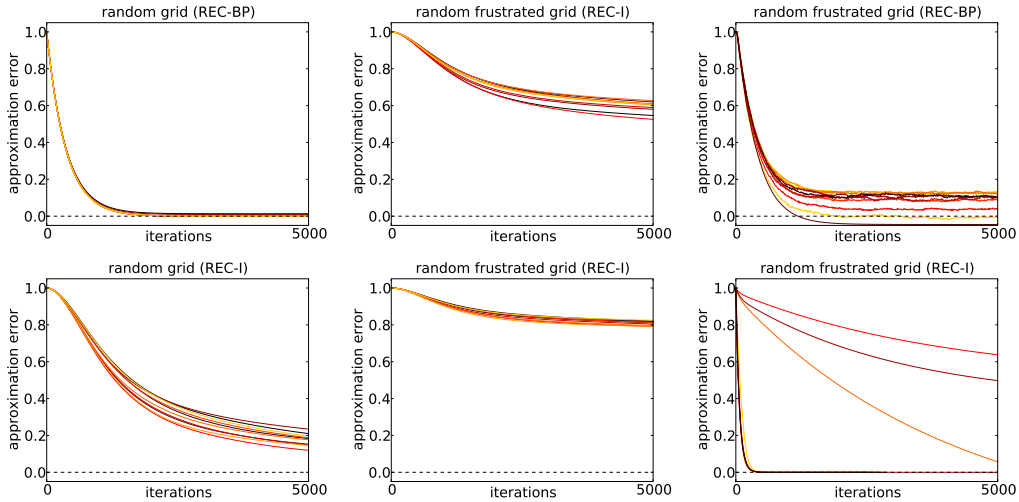

Figure 1: The REC algorithm in $10 \times 10$ grids. Left column: random grids, using REC-BP (top) and REC-I (bottom). Center column: frustrated grids, using REC-I with $p = \frac{1}{2}$ (top), $p = \frac{1}{3}$ (bottom). Right column: frustrated grids, using REC-BP (top) with a fully disconnected relaxation, and REC-I (bottom) with a relaxation with max cluster size 3.

## 6 Experiments

Our goal in this section is to highlight the degree to which different types of compensations can tighten a relaxation, as well as to highlight the differences in the iterative algorithms to find them. We evaluated our compensations using randomly parametrized $10 \times 10$ grid networks. We judge the quality of an approximation by the degree to which a compensation is able to improve a relaxation. In particular, we measured the error $E = \frac{\frac{1}{1+k}\text{c-map}^\star - \text{map}^\star}{\text{r-map}^\star - \text{map}^\star}$ which is zero when the compensation is exact, and one when the compensation is equivalent to the relaxation (remember that we initialize the REC algorithm, for both types of compensations, with parameters that led to an initial compensation with an optimal MAP value $\frac{1}{1+k}\text{c-map}_0^\star = \text{r-map}^\star$). Note also that we use no instances where the error $E$ is undefined, i.e., $\text{r-map}^\star - \text{c-map}^\star = 0$, where the relaxation alone was able to recover the exact solution.

We first consider grid networks where factors $\psi_a(x_i, x_j)$ were assigned to grid edges $(i, j)$, with values drawn uniformly at random from 0 to 1 (we assigned no factors to nodes). We assumed first the coarsest possible relaxation, one that results in a fully disconnected approximation, and where the MAP value is found by maximizing factors independently.[2] We expect a relaxation's upper bound to be quite loose in this case.

Consider first Figure 1 (left), where we generated ten random grid networks (we plotted only ten for clarity) and plotted the compensation errors ($y$-axis) as they evolved over iterations ($x$-axis). At iteration 0, the MAP value of each compensation is equivalent to that of the relaxation (by design). We see that, once we start iterating, that both methods of compensation can tighten the approximation of our very coarse relaxation. For REC-BP, we do so relatively quickly (in fewer iterations), and to exact or near-exact levels (note that the 10 instances plotted behave similarly). For REC-I, convergence is slower, but the compensation is still a significant improvement over the relaxation. Moreover, it is apparent that further iterations would benefit the compensation further.

We next generated random grid networks with frustrated interactions. In particular, each edge was given either an attractive factor or repulsive factor, at random each with probability $\frac{1}{2}$. An attractive factor $\psi_a(X_i, X_j)$ was given a value at random from $1 - p$ to 1 if $x_i = x_j$ and a value from 0 to

$p$ if $x_i \neq x_j$, which favors configurations $x_i = x_j$ when $p \leq \frac{1}{2}$. Similarly for repulsive factors, which favors instead configurations where $x_i \neq x_j$. It is well known that belief propagation tends to not converge in networks with frustrated interactions [11]. Non-convergence is the primary failure mode for belief propagation, and in such cases, we may try to use instead REC-I. We generated 10 random grid networks with $p = \frac{1}{2}$ and another 10 networks with $p = \frac{1}{3}$. Although the frustration in these networks is relatively mild, REC-BP did not converge in any of these cases. On the other hand, REC-I compensations were relatively well behaved, and produced monotonically decreasing upper bounds on the MAP value; see Figure 1 (center). Although the degree of compensation is not as dramatic, we note that we are compensating for a very coarse relaxation (fully disconnected).

In Figure 1 (right), we considered frustrated grid networks where $p = \frac{1}{10}$, where REC-BP converged in only one of 10 networks generated. Moreover, we can see in that one instance, REC-BP converges below the true MAP value; remember that by Theorem 3, REC-I compensations always yield upper bounds. In the case of REC-I, the compensations did not improve significantly on the fully disconnected relaxations (not shown). It is, however, straightforward to try less extreme relaxations. For example, we used the mini-buckets-based approach to relaxation proposed in [4], and identified relaxed models $\mathcal{M}'$ with jointrees that had a maximum cluster size of 3 (c.f., [12] which re-introduced constraints over triples). Surprisingly, this was enough for REC-I to compensate for the relaxation completely (to within $10^{-8}$) in 7 of the 10 instances plotted. REC-BP benefits from added structure as well, converging and compensating completely (to within $10^{-4}$) in 9 of 10 instances (not plotted).

## 7 Discussion

There are two basic concepts underlying our proposed framework. The first is to relax a problem by dropping equivalence constraints. The second is that of compensating for a relaxation in ways that can capture existing algorithms as special cases, and in ways that allow us to design new algorithms. The idea of using structural relaxations for upper-bounding MAP solutions in probabilistic graphical models goes back to mini-bucket approximations [13], which can be considered to be a particular way of relaxing equivalence constraints from a model [4]. In this paper, we propose further a way to compensate for these relaxations, by restoring a weaker notion of equivalence. One approach to compensation identified a generalized class of max-product belief propagation approximations. We then identified a second approach that led to another class of approximations that we have observed to yield tighter upper bounds on MAP solutions as compared to a relaxation alone.

An orthogonal approach to upper-bounding MAP solutions is based on linear programming (LP) relaxations, which has seen significant interest in recent years [1, 2]. This perspective is based on formulating MAP problems as integer programs, whose solutions are upper-bounded by tractable LP relaxations. A related approach based on Lagrangian relaxations is further capable of incorporating structural simplifications [14]. Indeed, there has been significant interest in identifying a precise connection between belief propagation and LP relaxations [2, 10].

In contrast to the above approaches, compensations further guarantee, in Theorem 4, upper bounds on MAP solutions under *any* partial assignment (without rerunning the algorithm). This property has the potential to impact algorithms, such as [3, 4], that rely on such upper bounds, under partial assignments, to perform a branch-and-bound search for optimal MAP solutions.[3] Further, as we approximate MAP by computing it exactly in a compensated model, we avoid the difficulties that algorithms such as max-product BP and related algorithms face, which infer MAP assignments using max-marginals (which may not have unique maximal states), which is based on local information only [1]. The perspective that we propose further allows us to identify the intuitive differences between belief propagation and an upper-bound approximation, namely that they arise from different notions of compensation. We hope that this perspective will enable the design of new approximations, especially in domains where specific notions of compensation may suggest themselves.

**Acknowledgments**

This work has been partially supported by NSF grant #IIS-0916161.

## Footnotes

[1]c-map$^\star_0 = \max_\mathbf{x} \text{c-map}_0(\mathbf{x}) = \max_\mathbf{x}[\text{r-map}(\mathbf{x}) + \sum_{X_i \equiv X_j} \theta(X_i\!=\!x) + \theta(X_j\!=\!x)]$

[2]For each factor $\psi_a$ and for each variable $X$ in $\psi_a$, we replaced variable $X$ with a unique clone $\hat{X}$ and introduced the equivalence constraint $X \equiv \hat{X}$. When we then relax all equivalence constraints, the resulting factor graph is fully disconnected. This corresponds to deleting all factor graph edges, as described in Section 3.

[3] We investigated the use of REC-I approximations in depth-first branch-and-bound search for solving weighted Max-SAT problems, where we were able to use a more specialized iterative algorithm [15].

# References

[1] Martin J. Wainwright, Tommi Jaakkola, and Alan S. Willsky. MAP estimation via agreement on trees: message-passing and linear programming. *IEEE Transactions on Information Theory*, 51(11):3697–3717, 2005.

[2] Amir Globerson and Tommi Jaakkola. Fixing max-product: Convergent message passing algorithms for MAP LP-relaxations. In *NIPS*, pages 553–560, 2008.

[3] Radu Marinescu, Kalev Kask, and Rina Dechter. Systematic vs. non-systematic algorithms for solving the MPE task. In *UAI*, pages 394–402, 2003.

[4] Arthur Choi, Mark Chavira, and Adnan Darwiche. Node splitting: A scheme for generating upper bounds in Bayesian networks. In *UAI*, pages 57–66, 2007.

[5] Rina Dechter and Irina Rish. Mini-buckets: A general scheme for bounded inference. *J. ACM*, 50(2):107–153, 2003.

[6] Arthur Choi and Adnan Darwiche. An edge deletion semantics for belief propagation and its practical impact on approximation quality. In *AAAI*, pages 1107–1114, 2006.

[7] Arthur Choi and Adnan Darwiche. Approximating the partition function by deleting and then correcting for model edges. In *UAI*, pages 79–87, 2008.

[8] Rina Dechter, Kalev Kask, and Robert Mateescu. Iterative join-graph propagation. In *UAI*, pages 128–136, 2002.

[9] Martin J. Wainwright, Tommi Jaakkola, and Alan S. Willsky. Tree consistency and bounds on the performance of the max-product algorithm and its generalizations. *Statistics and Computing*, 14:143–166, 2004.

[10] Yair Weiss, Chen Yanover, and Talya Meltzer. MAP estimation, linear programming and belief propagation with convex free energies. In *UAI*, 2007.

[11] Gal Elidan, Ian McGraw, and Daphne Koller. Residual belief propagation: Informed scheduling for asynchronous message passing. In *UAI*, 2006.

[12] David Sontag, Talya Meltzer, Amir Globerson, Tommi Jaakkola, and Yair Weiss. Tightening LP relaxations for MAP using message passing. In *UAI*, pages 503–510, 2008.

[13] Rina Dechter. Mini-buckets: a general scheme for approximation in automated reasoning. In *Proc. International Joint Conference on Artificial Intelligence (IJCAI)*, pages 1297–1302, 1997.

[14] Jason K. Johnson, Dmitry M. Malioutov, and Alan S. Willsky. Lagrangian relaxation for MAP estimation in graphical models. In *Proceedings of the 45th Allerton Conference on Communication, Control and Computing*, pages 672–681, 2007.

[15] Arthur Choi, Trevor Standley, and Adnan Darwiche. Approximating weighted Max-SAT problems by compensating for relaxations. In *Proceedings of the 15th International Conference on Principles and Practice of Constraint Programming (CP)*, pages 211–225, 2009.

